# Information Theoretic Analysis of Connection Structure from Spike Trains

**Satoru Shiono***
Central Research Laboratory
Mitsubishi Electric Corporation
Amagasaki, Hyogo 661, Japan

**Satoshi Yamada**
Central Research Laboratory
Mitsubishi Electric Corporation
Amagasaki, Hyogo 661, Japan

**Michio Nakashima**
Central Research Laboratory
Mitsubishi Electric Corporation
Amagasaki, Hyogo 661, Japan

**Kenji Matsumoto**
Faculty of Pharmaceutical Science
Hokkaidou University
Sapporo, Hokkaidou 060, Japan

## Abstract

We have attempted to use information theoretic quantities for analyzing neuronal connection structure from spike trains. Two point mutual information and its maximum value, channel capacity, between a pair of neurons were found to be useful for sensitive detection of crosscorrelation and for estimation of synaptic strength, respectively. Three point mutual information among three neurons could give their interconnection structure. Therefore, our information theoretic analysis was shown to be a very powerful technique for deducing neuronal connection structure. Some concrete examples of its application to simulated spike trains are presented.

## 1 INTRODUCTION

The deduction of neuronal connection structure from spike trains, including synaptic strength estimation, has long been one of the central issues for understanding the structure and function of the neuronal circuit and thus the information processing

---

mechanism at the neuronal circuitry level. A variety of crosscorrelational techniques for two or more neurons have been proposed and utilized (*e.g.*, Melssen and Epping, 1987; Aertsen *et. al.*, 1989). There are, however, some difficulties with those techniques, as discussed by, *e.g.*, Yang and Shamma (1990). It is sometimes difficult for the method to distinguish a significant crosscorrelation from noise, especially when the amount of experimental data is limited. The quantitative estimation of synaptic connectivity is another difficulty. And it is impossible to determine whether two neurons are directly connected or not, only by finding a significant crosscorrelation between them.

The information theory has been shown to afford a powerful tool for the description of neuronal input-output relations, such as in the investigation on the neuronal coding of the visual cortex (Eckhorn *et. al.*, 1976; Optican and Richmond, 1987). But there has been no extensive study to apply it to the correlational analysis of action potential trains. Because a correlational method using information theoretic quantities is considered to give a better correlational measure, the information theory is expected to offer a unique correlational method to overcome the above difficulties.

In this paper, we describe information theory-based correlational analysis for action potential trains, using two and three point mutual information (MI) and channel capacity. Because the information theoretic analysis by two point MI and channel capacity will be published in near future (Yamada *et. al.*, 1993a), more detailed description is given here on the analysis by three point MI for infering the relationship among three neurons.

## 2    CORRELATIONAL ANALYSIS BASED ON INFORMATION THEORY

### 2.1    INFORMATION THEORETIC QUANTITIES

According to the information theory, the $n$ point mutual information expresses the amount of information shared among $n$ processes (McGill, 1955). Let $X$, $Y$ and $Z$ be processes, and $t$ and $s$ be the time delays of $X$ and $Y$ from $Z$, respectively. Using Shannon entropies $H$, two point MI between $X$ and $Y$ and three point MI, are defined (Shannon, 1948; Ikeda *et. al.*, 1989):

$$\begin{align}
I(X_t : Y_s) &= H(X_t) + H(Y_s) - H(X_t, Y_s), \tag{1}\\
I(X_t : Y_s : Z) &= H(X_t) + H(Y_s) + H(Z) - H(X_t, Y_s)\\
&\quad - H(Y_s, Z) - H(Z, X_t) + H(X_t, Y_s, Z). \tag{2}
\end{align}$$

$I(X_t : Y_s : Z)$ is related to $I(X_t : Y_s)$ as follows:

$$I(X_t : Y_s : Z) = I(X_t : Y_s) - I(X_t : Y_s | Z), \tag{3}$$

where $I(X_t : Y_s | Z)$ means the two point conditional MI between $X$ and $Y$ if the state of $Z$ is given. On the other hand, channel capacity is given by ($\tau = s - t$),

$$CC(X : Y_\tau) = \max_{p(x_t)} I(X : Y_\tau). \tag{4}$$

We consider now $X$, $Y$ and $Z$ to be neurons whose spike activity has been measured.

Two point MI and two point conditional MI are obtained by $(i, j, k = 0, 1)$,

$$I(X : Y_r) = \sum_{i,j} p(y_{j,r}|x_i)p(x_i)\log\frac{p(y_{j,r}|x_i)}{p(y_{j,r})}, \tag{5}$$

$$I(X_t : Y_s|Z) = \sum_{i,j,k} p(x_{i,t}, y_{j,s}|z_k)p(z_k)\log\frac{p(x_{i,t}, y_{j,s}|z_k)}{p(x_{i,t}|z_k)p(y_{j,s}|z_k)}. \tag{6}$$

where $x$, $y$ and $z$ mean the states of neurons, e.g., $x_1$ for the firing state and $x_0$ for the non-firing state of $X$, and $p(\ )$ denotes probability. And three point MI is obtained by using Equation (3). Those information theoretic quantities are calculated by using the probabilities estimated from the spike trains of $X$, $Y$ and $Z$ after the spike trains are converted into time sequences consisting of 0 and 1 with discrete time steps, as described elswhere (Yamada et. al., 1993a).

## 2.2 PROCEDURE FOR THREE POINT MUTUAL INFORMATION ANALYSIS

Suppose that a three point MI peak is found at $(t_0, s_0)$ in the $t$, $s$-plane (see Figure 1). The three time delays, $t_0, s_0$ and $\tau = s_0 - t_0$, are obtained. They are supposed to be time delays in three possible interconnections between any pair of neurons. Because the peak is not significant if only one pair of the three neurons is interconnected, two or three of the possible interconnections with corresponding time delays should truly work to produce the peak. We will utilize $I(n : m)$ and $I(n : m|l)$ $(n, m, l = X, Y$ or $Z)$ at the peak to find working interconnections out of them. These quantities are obtained by recalculating each probability in Equations (5) and (6) over the whole peak region.

If two neurons, e.g., $X$ and $Y$, are not interconnected either $I(X : Y)$ or $I(X : Y|Z)$ is equal to zero. The reverse proposition, however, is not true. The necessary and sufficient condition for having no interconnection is obtained by calculating $I(n : m)$ and $I(n : m|l)$ for all possible interconnection structures. The neurons are rearranged and renamed $A, B$ and $C$ in the order of the time delays. There are only four interconnection structures, as shown in Table 1.

**I:** No interconnection between $A$ and $B$. $A$ and $B$ are statistically independent, i.e., $p(a_i, b_j) = p(a_i)p(b_j)$, $I(A : B) = 0$. The three point MI peak is negative.

**II:** No interconnection between $A$ and $C$. The states of $A$ and $C$ are statistically independent when the state of $B$ is given, i.e., $p(a_i, c_k|b_j) = p(a_i|b_j)p(c_k|b_j)$, $I(A : C|B) = 0$. The peak is positive.

**III:** No interconnection between $B$ and $C$. Similar to case II, because $p(b_j, c_k|a_i) = p(b_j|a_i)p(c_k|a_i)$, $I(B : C|A) = 0$. The peak is positive.

**IV:** Three interconnections. The above three cases are considered to occur concomitantly in this case. The peak is positive or negative, depending on their relative contributions. Because $A$ and $B$ should have an apparent effect on the firing-probability of the postsynaptic neurons, $I(A : B)$, $I(A : C|B)$ and $I(B : C|A)$ are all non-zero except for the case where the activity of $B$ completely coincides with that of $A$ with the specified time delay (in this case, both $I(A : C|B)$ and $I(B : C|A)$ are zero (see Yamada et. al., 1993b)).

Table 1. Interconnection Structure and Information Theoretic Quantities

| Interconnection Structure | I: 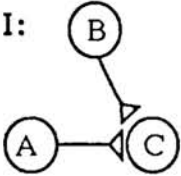 | II: 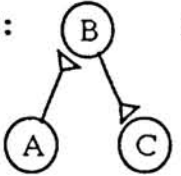 | III: 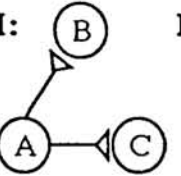 | IV: 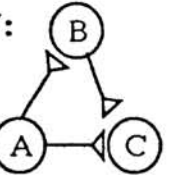 |
|---|---|---|---|---|
| **2 point MI** | | | | |
| $I(A{:}B)$ | $= 0$ | $> 0$ | $> 0$ | $> 0$ |
| $I(A{:}C)$ | $\geqq 0$ | $> 0$ | $> 0$ | $\geqq 0$ |
| $I(B{:}C)$ | $\geqq 0$ | $> 0$ | $> 0$ | $\geqq 0$ |
| **2 point condition MI** | | | | |
| $I(A{:}B\,|\,C)$ | $> 0$ | $\geqq 0$ | $\geqq 0$ | $\geqq 0$ |
| $I(A{:}C\,|\,B)$ | $> 0$ | $= 0$ | $\geqq 0$ | $> 0$ |
| $I(B{:}C\,|\,A)$ | $> 0$ | $\geqq 0$ | $= 0$ | $> 0$ |
| **3 point MI** | | | | |
| $I(A{:}B{:}C)$ | $-$ | $+$ | $+$ | $+$ or $-$ |

From what we have described above, the interconnection structure for a three point MI peak is deduced utilizing the following procedure;

(a) A negative 3pMI peak: it corresponds to case I or IV. The problem is to determine whether $A$ and $B$ are interconnected or not.

  (1) If $I(A : B) = 0$, case I.
  (2) If $I(A : B) > 0$, case IV.

(b) A positive 3pMI peak: it corresponds to case II, III or IV. The existence of the $A-C$ and $B-C$ interconnections has to be checked.

  (1) If $I(A : C|B) > 0$ and $I(B : C|A) > 0$, case IV.
  (2) If $I(A : C|B) = 0$ and $I(B : C|A) > 0$, case II.
  (3) If $I(A : C|B) > 0$ and $I(B : C|A) = 0$, case III.
  (4) If $I(A : C|B) = 0$ and $I(B : C|A) = 0$, the interconnection structure cannot be deduced except for the $A-B$ interconnection.

This procedure is applicable, if all the time delays are non-zero. If otherwise, some of the interconnections cannot be determined (Yamada *et. al.*, 1993b).

## 3   SIMULATED SPIKE TRAINS

In order to characterize our information theoretic analysis, simulations of neuronal network models were carried out. We used a model neuron described by

the Hodgkin-Huxley equations (Yamada *et. al.*, 1989). The used equations and parameters were described (Yamada *et. al.*, 1993a). The Hodgkin-Huxley equations were mathematically integrated by the Runge-Kutta-Gill technique.

# 4    RESULTS AND DISCUSSION

## 4.1    ANALYSIS BY TWO POINT MUTUAL INFORMATION AND CHANNEL CAPACITY

The performance was previously reported of the information theoretic analysis by two point MI and channel capacity (Yamada *et. al.*, 1993a).

Briefly, this anlytical method was compared with some conventional ones for both excitatory and inhibitory connections using action potential trains obtained by the simulation of a model neuronal network. It was shown to have the following advantages. First, it reduced correlational measures within the bounds of noise and simultaneously amplified beyond the bounds by its nonlinear function. It should be easier in its crosscorrelation graph to find a neuron pair having a weak but significant interaction, especially when the synaptic strength is small or the amount of experimental data is limited. Second, channel capacity was shown to allow fairly effective estimation of synaptic strength, being independent of the firing probability of a presynaptic neuron, as long as this firing probability was not large enough to have the overlap of two successive postsynaptic potentials.

## 4.2    ANALYSIS BY THREE POINT MUTUAL INFORMATION

The practical application of the analysis by three point MI is shown below in detail, using spike trains obtained by simulation of the three-neuron network models shown in Figures 1 and 2 (Yamada *et. al.*, 1993b).

The network model in Figure 1(1) has three interconnections. In Figure 1(2), three point MI has two positive peaks at (17ms, 12ms) (unit "ms" is omitted hereafter) and $(17, 30)$, and one negative peak at $(0, 12)$. For the peak at $(17, 12)$, the neurons are renamed $A$, $B$ and $C$ from the time delays ($Z$ as $A$, $Y$ as $B$ and $X$ as $C$), as in Table 1. Because only $I(B : C|A) \doteq 0$ (see Figure 1 legend), the peak indicates case III with $A{\to}B$ ($Z{\to}Y$) ($s = 12$) and $A{\to}C$ ($Z{\to}X$) ($t = 17$) interconnections. Similarly, the peak at $(17, 30)$ indicates $Z{\to}X$ and $X{\to}Y$ ($s - t = 13$) interconnections, and the peak at $(0, 12)$ indicates $Z{\to}Y$ and $X{\to}Y$ interconnections. The interconnection structure deduced from each three point MI peak is consistent with each other, and in agreement with the network model.

Alternatively, the three point MI graphical presentation such as shown in Figure 1(2) itself gives indication of some truly existing interconnections. If more than two three point MI peaks are found on one of the three lines, $t = t_0$, $s = s_0$ and $s-t = \tau_0$, the interconnection with the time delay represented by this line is considered to be real. For example, because the peaks at $(17, 12)$ and $(17, 30)$ are on the line of $t = 17$ (Figure 1(2)), the interconnection represented by $t = 17$ ($Z{\to}X$) are considered to be real. In a similar manner, the interconnections of $s = 12$ ($Z{\to}Y$) and $s - t = 12$ ($X{\to}Y$) are obtained. But this graphical indication is not complete, and thus the calculation of two point MI's and two point conditional MI's should be always

(1)

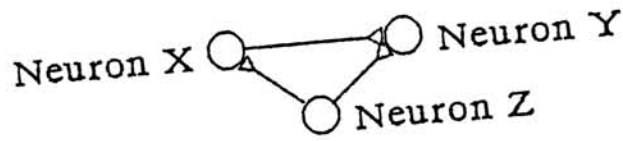

(2)

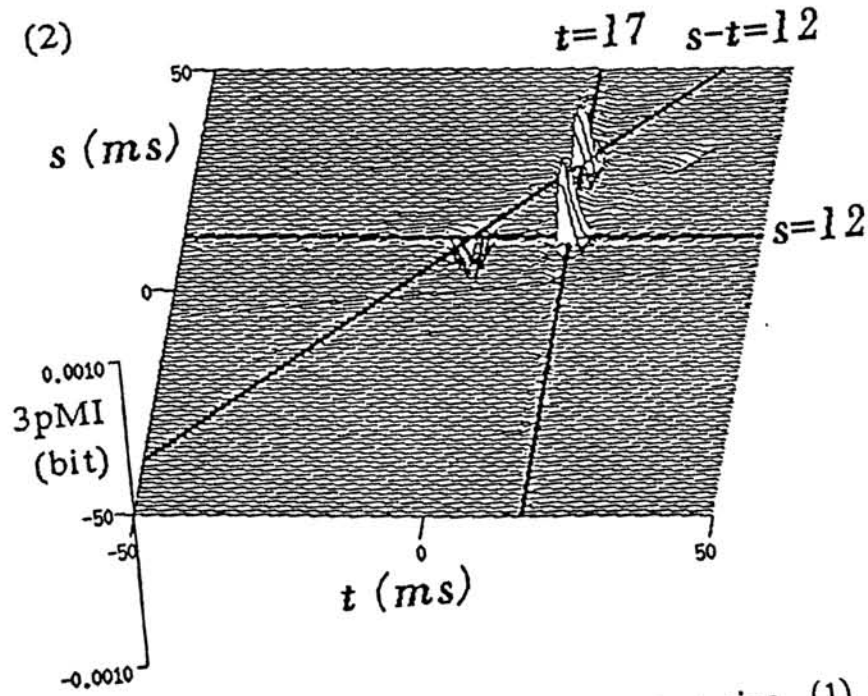

Figure 1. Three point MI analysis of simulated spike trains. (1) A three-neuron network model with $Z \to X$ $Z \to Y$ and $X \to Y$ interconnections. The total number of spikes; $X$:4000, $Y$:5400, $Z$:3150. (2) Three point MI analysis of spike trains. Three point MI has two positive peaks at $(17, 12)$ and $(17, 30)$, and one negative peak at $(0, 12)$. For the peak at $(17, 12)$ the neurons are renamed ($Z$ as $A$, $Y$ as $B$ and $X$ as $C$). Two point MI and two point conditional MI for the peak at $(17, 12)$ are: $I(A:B) = 0.03596$, $I(A:C) = 0.06855$, $I(B:C) = 0.01375$, $I(A:B|C) = 0.02126$, $I(A:C|B) = 0.05376$, $I(B:C|A) = 0.00011$. So, $I(B:C|A) \doteq 0$, indicating case III (see Table 1) with $A \to B$ ($Z \to Y$) and $A \to C$ ($Z \to X$) interconnections. Similarly, for the peaks at $(17, 30)$ and at $(0, 12)$, $Z \to X$ and $X \to Y$ interconnections, and $Z \to Y$ and $X \to Y$ interconnections are obtained, respectively.

performed for confirmation.

The network model in Figure 2(1) has four interconnections. Three point MI has five major peaks: four positive peaks at $(17, -12)$, $(17, 30)$, $(-24, -12)$ and $(17, 12)$ and one negative peak at $(0, 10)$. The peaks at $(17, -12)$, $(17, 12)$ and $(17, 30)$ are on the line of $t = 17$ ($Z \to X$), the peaks at $(17, -12)$ and $(-24, -12)$ are on the line of $s = 12$ line of $s = -12$ ($Z \leftarrow Y$), the peaks at $(17, 12)$ and $(0, 10)$ are on the line of $s = 12$ ($Z \to Y$), and the peaks at $(-24, -12)$, $(0, 10)$ and $(17, 30)$ are on the line of

(1)

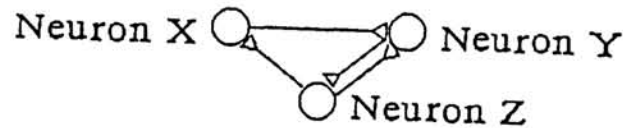

(2)

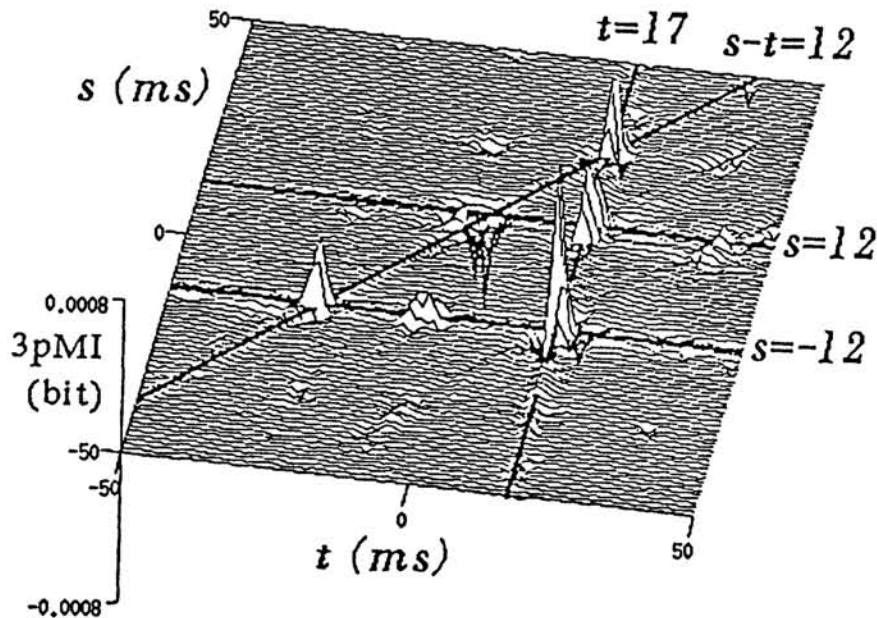

Figure 2. Three point MI analysis of simulated spike trains. (1) A three-neuron network model with $Z{\to}X$ $Z{\to}Y$, $Z{\leftarrow}Y$ and $X{\to}Y$ interconnections. The total number of spikes; $X$:4300, $Y$:5150, $Z$:4850. (2) Three point MI analysis of spike trains. Three point MI has five major peaks, four positive peaks at $(17, -12)$, $(17, 12)$, $(17, 30)$ and $(-24, -12)$, and one negative peak at $(0, 10)$.

$s - t = 12$ $(X{\to}Y)$. The calculation of two point MI and two point conditional MI for each peak gives the confirmation that each three point MI peak was produced by two interconnections. Namely, their calculation indicates $Z{\to}X$ $(t = 17)$, $Z{\leftarrow}Y$ $(s = -12)$, $Z{\to}Y$ $(s = 12)$ and $X{\to}Y$ $(s - t = 12)$ interconnections. There are also some small peaks. They are considered to be ghost peaks due to two or three interconnections, at least one of which is a combination of two interconnections found by analyzing the major peaks. For example, the positive peak at $(-7, -12)$ indicates $Z{\leftarrow}Y$ and $X{\to}Y$ interconnections, but the latter $(s - t = -5)$ is the combination of the $Z{\to}X$ interconnection $(t = 17)$ and the $Z{\to}Y$ interconnection $(s = 12)$.

The interconnection structure of a network containing an inhibitory interconnection or consisting of more than four neurons can also be deduced, although it becomes more difficult to perform the three point MI analysis.

## References

A. M. H. J. Aertsen, G. L. Gerstein, M. K. Habib & G. Palm. (1989) Dynamics of neuronal firing correlation: modulation of "effective connectivity". *J. Neurophysiol.* **61**: 900-917.

R. Eckhorn, O. J. Grüsser, J. Kröller, K. Pellnitz & B. Pöpel. (1976) Efficiency of different neuronal codes: information transfer calculations for three different neuronal systems. *Biol. Cybern.* **22**: 49-60.

K. Ikeda, K. Otsuka & K. Matsumoto. (1989) Maxwell-Bloch turbulence. *Prog. Theor. Phys., Suppl.* **99**: 295-324.

W. J. McGill. (1955) Multivariate information transmission. *IRE Trans. Inf. Theory* **1**: 93-111.

W. J. Melssen & W. J. M. Epping. (1987) Detection and estimation of neural connectivity based on crosscorrelation analysis. *Biol. Cybern.* **57**: 403-414.

L. M. Optican & B. J. Richmond. (1987) Temporal encoding of two-dimensional patterns by single units primate inferior temporal cortex. III. Information theoretic analysis. *J. Neurophysiol.* **57**: 162-178.

C. E. Shannon. (1948) A mathematical theory of communication. *Bell. Syst. Techn. J.* **27**: 379-423.

S. Yamada, M. Nakashima, K. Matsumoto & S. Shiono. (1993a) Information theoretic analysis of action potential trains: I. Analysis of correlation between two neurons. *Biol. Cybern.*, in press.

S. Yamada, M. Nakashima, K. Matsumoto & S. Shiono. (1993b) Information theoretic analysis of action potential trains: II. Analysis of correlation among three neurons. submitted to *Biol. Cybern.*

W. M. Yamada, C. Koch & P. R. Adams. (1989) Multiple channels and calcium dynamics. In C. Koch & I. Segev (ed), *Methods in Neuronal Modeling: From Synapses to Neurons*, 97-133, Cambridge, MA, USA: MIT Press.

X. Yang & S. A. Shamma. (1990) Identification of connectivity in neural networks. *Biophys. J.* **57**: 987-999.
